# A Learning Framework for Nearest Neighbor Search

**Lawrence Cayton**
Department of Computer Science
University of California, San Diego
lcayton@cs.ucsd.edu

**Sanjoy Dasgupta**
Department of Computer Science
University of California, San Diego
dasgupta@cs.ucsd.edu

## Abstract

Can we leverage learning techniques to build a fast nearest-neighbor (ANN) retrieval data structure? We present a general learning framework for the NN problem in which sample queries are used to learn the parameters of a data structure that minimize the retrieval time and/or the miss rate. We explore the potential of this novel framework through two popular NN data structures: KD-trees and the rectilinear structures employed by locality sensitive hashing. We derive a generalization theory for these data structure classes and present simple learning algorithms for both. Experimental results reveal that learning often improves on the already strong performance of these data structures.

## 1 Introduction

Nearest neighbor (NN) searching is a fundamental operation in machine learning, databases, signal processing, and a variety of other disciplines. We have a database of points $X = \{x_1, \ldots, x_n\}$, and on an input query $q$, we hope to return the nearest (or approximately nearest, or $k$-nearest) point(s) to $q$ in $X$ using some similarity measure.

A tremendous amount of research has been devoted to designing data structures for fast NN retrieval. Most of these structures are based on some clever partitioning of the space and a few have bounds (typically worst-case) on the number of distance calculations necessary to query it.

In this work, we propose a novel approach to building an efficient NN data structure based on *learning*. In contrast to the various data structures built using geometric intuitions, this learning framework allows one to construct a data structure by directly minimizing the cost of querying it.

In our framework, a sample query set guides the construction of the data structure containing the database. In the absence of a sample query set, the database itself may be used as a reasonable prior. The problem of building a NN data structure can then be cast as a learning problem:

> Learn a data structure that yields efficient retrieval times on the sample queries and is simple enough to generalize well.

A major benefit of this framework is that one can seamlessly handle situations where the query distribution is substantially different from the distribution of the database.

We consider two different function classes that have performed well in NN searching: KD-trees and the cell structures employed by locality sensitive hashing. The known algorithms for these data structures do not, of course, use learning to choose the parameters. Nevertheless, we can examine the generalization properties of a data structure *learned* from one of these classes. We derive generalization bounds for both of these classes in this paper.

Can the framework be practically applied? We present very simple learning algorithms for both of these data structure classes that exhibit improved performance over their standard counterparts.

## 2  Related work

There is a voluminous literature on data structures for nearest neighbor search, spanning several academic communities. Work on efficient NN data structures can be classified according to two criteria: whether they return exact or approximate answers to queries; and whether they merely assume the distance function is a metric or make a stronger assumption (usually that the data are Euclidean). The framework we describe in this paper applies to all these methods, though we focus in particular on data structures for $\mathbb{R}^D$.

Perhaps the most popular data structure for nearest neighbor search in $\mathbb{R}^D$ is the simple and convenient *KD-tree* [1], which has enjoyed success in a vast range of applications. Its main downside is that its performance is widely believed to degrade rapidly with increasing dimension. Variants of the data structure have been developed to ameliorate this and other problems [2], though high-dimensional databases continue to be challenging. One recent line of work suggests randomly projecting points in the database down to a low-dimensional space, and then using KD-trees [3, 4].

Locality sensitive hashing (LSH) has emerged as a promising option for high-dimensional NN search in $\mathbb{R}^D$ [5]. It has strong theoretical guarantees for databases of arbitrary dimensionality, though they are for *approximate* NN search. We review both KD-trees and LSH in detail later.

For data in metric spaces, there are several schemes based on repeatedly applying the triangle inequality to eliminate portions of the space from consideration; these include Orchard's algorithm [6] and AESA [7]. Metric trees [8] and the recently suggested spill trees [3] are based on similar ideas and are related to KD-trees. A recent trend is to look for data structures that are attuned to the *intrinsic dimension*, *e.g.* [9]. See the excellent survey [10] for more information.

There has been some work on building a data structure for a particular query distribution [11]; this line of work is perhaps most similar to ours. Indeed, we discovered at the time of press that the algorithm for KD-trees we describe appeared previously in [12]. Nevertheless, the learning theoretic approach in this paper is novel; the study of NN data structures through the lens of generalization ability provides a fundamentally different theoretical basis for NN search with important practical implications.

## 3  Learning framework

In this section we formalize a learning framework for NN search. This framework is quite general and will hopefully be of use to algorithmic developments in NN searching beyond those presented in this paper.

Let $X = \{x_1, \ldots, x_n\}$ denote the database and $\mathcal{Q}$ the space from which queries are drawn. A typical example is $X \subset \mathbb{R}^D$ and $\mathcal{Q} = \mathbb{R}^D$. We take a nearest neighbor data structure to be a mapping $f : \mathcal{Q} \to 2^X$; the interpretation is we compute distances only to $f(q)$, not all of $X$. For example, the structure underlying LSH partitions $\mathbb{R}^D$ into cells and a query is assigned to the subset of $X$ that falls into the same cell.

What quantities are we interested in optimizing? We want to only compute distances to a small fraction of the database on a query; and, in the case of probabilistic algorithms, we want a high probability of success. More precisely, we hope to minimize the following two quantities for a data structure $f$:

- The fraction of $X$ that we need to compute distances to:

$$\text{size}_f(q) \equiv \frac{|f(q)|}{n}.$$

- The fraction of a query's $k$ nearest neighbors that are missed:

$$\text{miss}_f(q) \equiv \frac{|\Gamma_k(q) \setminus f(q)|}{k}$$

($\Gamma_k(q)$ denotes the $k$ nearest neighbors of $q$ in $X$).

In $\epsilon$-approximate NN search, we only require a point $x$ such that $d(q, x) \le (1 + \epsilon)d(q, X)$, so we instead use an approximate miss rate:

$$\epsilon\mathrm{miss}_f(q) \equiv \mathbf{1}\left[\nexists x \in f(q) \text{ such that } d(q, x) \le (1 + \epsilon)d(q, X)\right].$$

None of the previously discussed data structures are built by explicitly minimizing these quantities, though there are known bounds for some. Why not? One reason is that research has typically focused on *worst-case* $\mathrm{size}_f$ and $\mathrm{miss}_f$ rates, which require minimizing these functions over *all* $q \in \mathcal{Q}$. $\mathcal{Q}$ is typically infinite of course.

In this work, we instead focus on *average-case* $\mathrm{size}_f$ and $\mathrm{miss}_f$ rates—*i.e.* we assume $q$ is a draw from some unknown distribution $\mathcal{D}$ on $\mathcal{Q}$ and hope to minimize

$$\mathbb{E}_{q \sim \mathcal{D}}\left[\mathrm{size}_f(q)\right] \quad \text{and} \quad \mathbb{E}_{q \sim \mathcal{D}}\left[\mathrm{miss}_f(q)\right].$$

To do so, we assume that we are given a sample query set $Q = \{q_1, \ldots, q_m\}$ drawn iid from $\mathcal{D}$. We attempt to build $f$ minimizing the empirical size and miss rates, then resort to generalization bounds to relate these rates to the true ones.

## 4 Learning algorithms

We propose two learning algorithms in this section. The first is based on a splitting rule for KD-trees designed to minimize a greedy surrogate for the empirical $\mathrm{size}_f$ function. The second is a algorithm that determines the boundary locations of the cell structure used in LSH that minimize a tradeoff of the empirical $\mathrm{size}_f$ and $\epsilon\mathrm{miss}_f$ functions.

### 4.1 KD-trees

KD-trees are a popular cell partitioning scheme for $\mathbb{R}^D$ based on the binary search paradigm. The data structure is built by picking a dimension, splitting the database along the median value in that dimension, and then recursing on both halves.

**procedure** BUILDTREE($S$)
  **if** $|S| <$ *MinSize*, **return** leaf.
  **else**:
    Pick an axis $i$.
    Let *median* $=$ median($s_i : s \in S$).
    *LeftTree* $=$ BUILDTREE($\{s \in S : s_i \le median\}$).
    *RightTree* $=$ BUILDTREE($\{s \in S : s_i > median\}$).
    **return** [*LeftTree*, *RightTree*, *median*, $i$].

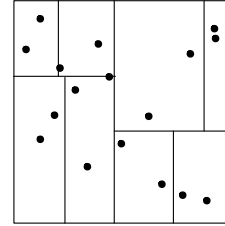

To find a NN for a query $q$, one first computes distances to all points in the same cell, then traverses up the tree. At each parent node, the minimum distance between $q$ and points already explored is compared to the distance to the split. If the latter is smaller, then the other child must be explored.

Explore right subtree: 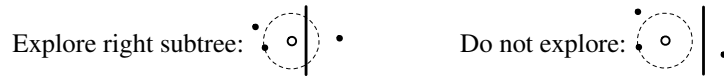 Do not explore:

Typically the cells contain only a few points; a query is expensive because it lies close to many of the cell boundaries and much of the tree must be explored.

**Learning method**

Rather than picking the median split at each level, we use the training queries $q_i$ to pick a split that greedily minimizes the expected cost. A split $s$ divides the sample queries (that are in the cell being split) into three sets: $Q_{tc}$, those $q$ that are "too close" to $s$—*i.e.* nearer to $s$ than $d(q, X)$; $Q_r$, those on the right of $s$ but not in $Q_{tc}$; and $Q_l$, those on the left of $s$ but not in $Q_{tc}$. Queries in $Q_{tc}$ will require exploring both sides of the split. The split also divides the database points (that are in the cell being split) into $X_l$ and $X_r$. The cost of split $s$ is then defined to be

$$\mathrm{cost}(s) \equiv |Q_l| \cdot |X_l| + |Q_r| \cdot |X_r| + |Q_{tc}| \cdot |X|.$$

cost$(s)$ is a greedy surrogate for $\sum_i \text{size}_f(q_i)$; evaluating the true average size would require a potentially costly recursion. In contrast, minimizing cost$(s)$ can be done painlessly since it takes on at most $2m + n$ possible values and each can be evaluated quickly. Using a sample set led us to a very simple, natural cost function that can be used to pick splits in a principled manner.

## 4.2 Locality sensitive hashing

LSH was a tremendous breakthrough in NN search as it led to data structures with provably sublinear (in the database size) retrieval time for approximate NN searching. More impressive still, the bounds on retrieval are independent of the dimensionality of the database. We focus on the LSH scheme for the $\|\cdot\|_p$ norm ($p \in (0, 2]$), which we refer to as $\text{LSH}_p$. It is built on an extremely simple space partitioning scheme which we refer to as a *rectilinear cell structure* (RCS).

**procedure** BUILDRCS$(X \subset \mathbb{R}^D)$

Let $R \in \mathbb{R}^{O(\log n) \times d}$ with $R_{ij}$ iid draws from a $p$-stable distribution.[1]

Project database down to $O(\log n)$ dimensions: $x_i \mapsto R x_i$.

Uniformly grid the space with $B$ bins per direction.

See figure 3, left panel, for an example. On query $q$, one simply finds the cell that $q$ belongs to, and returns the nearest $x$ in that cell.

In general, $\text{LSH}_p$ requires many RCSs, used in parallel, to achieve a constant probability of success; in many situations one may suffice [13]. Note that $\text{LSH}_p$ only works for distances at a single scale $R$: the specific guarantee is that $\text{LSH}_p$ will return a point $x \in X$ within distance $(1 + \epsilon)R$ of $q$ as long as $d(q, X) < R$. To solve the standard $\epsilon$ approximate NN problem, one must build $O(\log(n/\epsilon))$ $\text{LSH}_p$ structures.

### Learning method

We apply our learning framework directly to the class of RCSs since they are the core structural component of $\text{LSH}_p$. We consider a slightly wider class of RCSs where the bin widths are allowed to vary. Doing so potentially allows a *single* RCS to work at multiple scales if the bin positions are chosen appropriately. We give a simple procedure that selects the bin boundary locations.

We wish to select boundary locations minimizing the cost $\sum_i \epsilon\text{miss}_f(q_i) + \lambda\text{size}_f(q_i)$, where $\lambda$ is a tradeoff parameter (alternatively, one could fix a miss rate that is reasonable, say 5%, and minimize the size). The optimization is performed along one dimension at a time. Fortunately, the optimal binning along a dimension can be found by dynamic programming. There are at most $m+n$ possible boundary locations; order them from left to right. The cost of placing the boundaries at $p_1, p_2, p_{B+1}$ can be decomposed as $c[p_1, p_2] + \cdots + c[p_B, p_{B+1}]$, where

$$c[p_i, p_{i+1}] = \sum_{q \in [p_i, p_{i+1}]} \epsilon\text{miss}_f(q) + \lambda \sum_{q \in [p_i, p_{i+1}]} |\{x \in [p_i, p_{i+1}]\}|.$$

Let $D$ be our dynamic programming table where $D[p, i]$ is defined as the cost of putting the $i$th boundary at position $p$ and the remaining $B + 1 - i$ to the right. Then $D[p, i] = \min_{p' \geq p} c[p, p'] + D[p', i - 1]$.

## 5 Generalization theory[2]

In our framework, a nearest neighbor data structure is learned by specifically designing it to perform well on a set of sample queries. Under what conditions will this search structure have good performance on future queries?

Recall the setting: there is a database $X = \{x_1, \ldots, x_n\}$, sample queries $Q = \{q_1, \ldots, q_m\}$ drawn iid from some distribution $\mathcal{D}$ on $\mathcal{Q}$, and we wish to learn a data structure $f : \mathcal{Q} \to 2^X$ drawn from a

function class $\mathcal{F}$. We are interested in the generalization of $\text{size}_f(q) \equiv \frac{|f(q)|}{n}$, and $\text{miss}_f(q) \equiv \frac{|\Gamma_k(q)\setminus f(q)|}{k}$, both of which have range $[0,1]$ ($\epsilon\text{miss}_f(q)$ can be substituted for $\text{miss}_f(q)$ throughout this section).

Suppose a data structure $f$ is chosen from some class $\mathcal{F}$, so as to have low empirical cost

$$\frac{1}{m}\sum_{i=1}^{m}\text{size}_f(q_i) \quad \text{and} \quad \frac{1}{m}\sum_{i=1}^{m}\text{miss}_f(q_i).$$

Can we then conclude that data structure $f$ will continue to perform well for subsequent queries drawn from the underlying distribution on $\mathcal{Q}$? In other words, are the empirical estimates above necessarily close to the true expected values $\mathbb{E}_{q\sim\mathcal{D}}\text{size}_f(q)$ and $\mathbb{E}_{q\sim\mathcal{D}}\text{miss}_f(q)$ ?

There is a wide range of uniform convergence results which relate the difference between empirical and true expectations to the number of samples seen (in our case, $m$) and some measure of the complexity of the two classes $\{\text{size}_f : f \in \mathcal{F}\}$ and $\{\text{miss}_f : f \in \mathcal{F}\}$. The following is particularly convenient to use, and is well-known [14, theorem 3.2].

**Theorem 1.** *Let $\mathcal{G}$ be a set of functions from a set $\mathcal{Z}$ to $[0,1]$. Suppose a sample $z_1,\ldots,z_m$ is drawn from some underlying distribution on $\mathcal{Z}$. Let $\mathcal{G}_m$ denote the restriction of $\mathcal{G}$ to these samples, that is,*

$$\mathcal{G}_m = \{(g(z_1),g(z_2),\ldots,g(z_m)) : g \in \mathcal{G}\}.$$

*Then for any $\delta > 0$, the following holds with probability at least $1-\delta$:*

$$\sup_{g\in\mathcal{G}}\left|\mathbb{E}g - \frac{1}{m}\sum_{i=1}^{m}g(z_i)\right| \leq 2\sqrt{\frac{2\log|\mathcal{G}_m|}{m}} + \sqrt{\frac{\log(2/\delta)}{m}}.$$

This can be applied immediately to the kind of data structure used by LSH.

**Definition 2.** *A $(u_1,\ldots,u_d,B)$-rectilinear cell structure (RCS) in $\mathbb{R}^D$ is a partition of $\mathbb{R}^D$ into $B^d$ cells given by*

$$x \mapsto (h_1(x\cdot u_1),\ldots,h_d(x\cdot u_d)),$$

*where each $h_i : \mathbb{R} \to \{1,\ldots,B\}$ is a partition of the real line into $B$ intervals.*

**Theorem 3.** *Fix any vectors $u_1,\ldots,u_d \in \mathbb{R}^D$, and, for some positive integer $B$, let the set of data structures $\mathcal{F}$ consist of all $(u_1,\ldots,u_d,B)$-rectilinear cell structures in $\mathbb{R}^D$. Fix any database of $n$ points $X \subset \mathbb{R}^D$. Suppose there is an underlying distribution over queries in $\mathbb{R}^D$, from which $m$ sample queries $q_1,\ldots,q_m$ are drawn. Then*

$$\sup_{f\in\mathcal{F}}\left|\mathbb{E}[\text{miss}_f] - \frac{1}{m}\sum_{i=1}^{m}\text{miss}_f(q_i)\right| \leq 2\sqrt{\frac{2d(B-1)\log(m+n)}{m}} + \sqrt{\frac{\log(2/\delta)}{m}}$$

*and likewise for $\text{size}_f$.*

*Proof.* Fix any $X = \{x_1,\ldots,x_n\}$ and any $q_1,\ldots,q_m$. In how many ways can these points be assigned to cells by the class of all $(u_1,\ldots,u_d,B)$-rectilinear data structures? Along each axis $u_i$ there are $B-1$ boundaries to be chosen and only $m+n$ distinct locations for each of these (as far as partitioning of the $x_i$'s and $q_i$'s is concerned). Therefore there are at most $(m+n)^{d(B-1)}$ ways to carve up the points. Thus the functions $\{\text{miss}_f : f \in \mathcal{F}\}$ (or likewise, $\{\text{size}_f : f \in \mathcal{F}\}$) collapse to a set of size just $(m+n)^{d(B-1)}$ when restricted to $m$ queries; the rest follows from theorem 1. $\square$

This is good generalization performance because it depends only on the *projected* dimension, not the original dimension. It holds when the projection directions $u_1,\ldots,u_d$ are chosen randomly, but, more remarkably, even if they are chosen based on $X$ (for instance, by running PCA on $X$). If we learn the projections as well (instead of using random ones) the bound degrades substantially.

**Theorem 4.** *Consider the same setting as Theorem 3, except that now $\mathcal{F}$ ranges over $(u_1,\ldots,u_d,B)$-rectilinear cell structures for all choices of $u_1,\ldots,u_d \in \mathbb{R}^D$. Then with probability at least $1-\delta$,*

$$\sup_{f\in\mathcal{F}}\left|\mathbb{E}[\text{miss}_f] - \frac{1}{m}\sum_{i=1}^{m}\text{miss}_f(q_i)\right| \leq 2\sqrt{\frac{2+2d(D+B-2)\log(m+n)}{m}} + \sqrt{\frac{\log(2/\delta)}{m}}$$

*and likewise for $\text{size}_f$.*

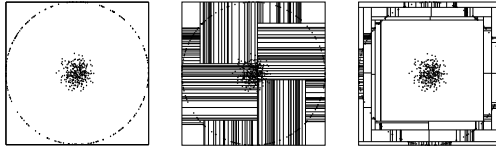

Figure 1: **Left**: Outer ring is the database; inner cluster of points are the queries. **Center**: KD-tree with standard median splits. **Right**: KD-tree with learned splits.

KD-trees are slightly different than RCSs: the directions $u_i$ are simply the coordinate axes, and the number of partitions per direction varies (*e.g.* one direction may have 10 partitions, another only 1).

**Theorem 5.** *Let $\mathcal{F}$ be the set of all depth $\eta$ KD-trees in $\mathbb{R}^D$ and $X \subset \mathbb{R}^D$ be a database of points. Suppose there is an underlying distribution over queries in $\mathbb{R}^D$ from which $q_1, \ldots q_m$ are drawn. Then with probability at least $1 - \delta$,*

$$\sup_{f \in \mathcal{F}} \left| \mathbb{E}[\text{miss}_f] - \frac{1}{m} \sum_{i=1}^{m} \text{miss}_f(q_i) \right| \leq 2\sqrt{\frac{(2^{\eta+1} - 2) \log\left(D(3m + n)\right)}{m}} + \sqrt{\frac{\log\left(2/\delta\right)}{m}}$$

A KD-tree utilizing median splits has depth $\eta \leq \log n$. The depth of a KD-tree with learned splits can be higher, though we found empirically that the depth was always much less than $2 \log n$ (and can of course be restricted manually). KD-trees require significantly more samples than RCSs to generalize; the class of KD-trees is much more complex than that of RCSs.

# 6 Experiments[3]

## 6.1 KD-trees

First let us look at a simple example comparing the learned splits to median splits. Figure 1 shows a 2-dimensional dataset and the cell partitions produced by the learned splits and the median splits. The KD-tree constructed with the median splitting rule places nearly all of the boundaries running right through the queries. As a result, nearly the entire database will have to be searched for queries drawn from the center cluster distribution. The KD-tree with the learned splits places most of the boundaries right around the actual database points, ensuring that fewer leaves will need to be examined for each query.

We now show results on several datasets from the UCI repository and 2004 KDD cup competition. We restrict attention to relatively low-dimensional datasets ($D < 100$) since that is the domain in which KD-trees are typically applied. These experiments were all conducted using a modified version of Mount and Arya's excellent KD-tree software [15]. For this set of experiments, we used a randomly selected subset of the dataset as the database and a separate small subset as the test queries. For the sample queries, we used the database itself—*i.e. no additional data was used to build the learned KD-tree*.

The following table shows the results. We compare performance in terms of the average number of database points we have to compute distances to on a test set.

| data set | DB size | test pts | dim | # distance calculations | | % |
| --- | --- | --- | --- | --- | --- | --- |
| | | | | median split | learned split | improvement |
| Corel (UCI) | 32k | 5k | 32 | 1035.7 | 403.7 | 61.0 |
| Covertype (UCI) | 100k | 10k | 55 | 20.8 | 18.4 | 11.4 |
| Letter (UCI) | 18k | 2k | 16 | 470.1 | 353.8 | 27.4 |
| Pen digits (UCI) | 9k | 1k | 16 | 168.9 | 114.9 | 31.9 |
| Bio (KDD) | 100k | 10k | 74 | 1409.8 | 1310.8 | 7.0 |
| Physics (KDD) | 100k | 10k | 78 | 1676.6 | 404.0 | 75.9 |

The learned method outperforms the standard method on all of the datasets, showing a very large improvement on several of them. Note also that even the standard method exhibits good performance,

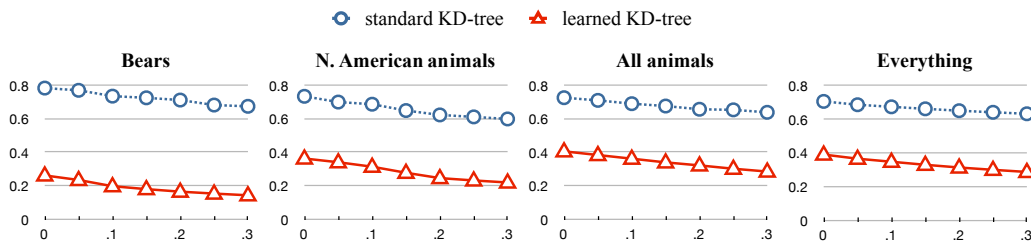

Figure 2: Percentage of DB examined as a function of (the approximation factor) for various query distributions.

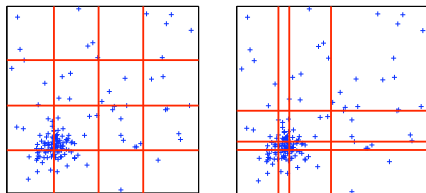

Figure 3: Example RCSs. **Left**: Standard RCS. **Right**: Learned RCS

often requiring distance calculations to less than one percent of the database. We are showing strong improvements on what are already quite good results.

We additionally experimented with the 'Corel50' image dataset. It is divided into 50 classes (*e.g.* air shows, bears, tigers, Fiji) containing 100 images each. We used the 371-dimensional "semantic space" representation of the images recently developed in a series of image retrieval papers (see *e.g.* [16]). This dataset allows us to explore the effect of differing query and database distributions in a natural setting. It also demonstrates that KD-trees with learned parameters can perform well on high-dimensional data.

Figure 2 shows the results of running KD-trees using median and learned splits. In each case, 4000 images were chosen for the database (from across all the classes) and images from select classes were chosen for the queries. The "All" queries were chosen from all classes; the "Animals" were chosen from the 11 animal classes; the "N. American animals" were chosen from 5 of the animal classes; and the "Bears" were chosen from the two bear classes. Standard KD-trees are performing somewhat better than brute force in these experiments; the learned KD-trees yield much faster retrieval times across a range of approximation errors. Note also that the performance of the learned KD-tree seems to improve as the query distribution becomes simpler whereas the performance for the standard KD-tree actually degrades.

## 6.2 RCS/LSH

Figure 3 shows a sample run of the learning algorithm. The queries and DB are drawn from the same distribution. The learning algorithm adjusts the bin boundaries to the regions of density.

Experimenting with RCS structures is somewhat challenging since there are two parameters to set (number of projections and boundaries), an approximation factor, and two quantities to compare (size and miss). We swept over the two parameters to get results for the standard RCSs. Results for learned RCSs were obtained using only a single (essentially unoptimized) parameter setting. Rather than minimizing a tradeoff between $size_f$ and $miss_f$, we constrained the miss rate and optimized the $size_f$. The constraint was varied between runs (2%, 4%, *etc.*) to get comparable results.

Figure 4 shows the comparison on databases of 10k points drawn from the MNIST and Physics datasets (2.5k points were used as sample queries). We see a marked improvement for the Physics dataset and a small improvement for the MNIST dataset. We suspect that the learning algorithm helps substantially for the physics data because the one-dimensional projections are highly non-uniform whereas the MNIST one-dimensional projections are much more uniform.

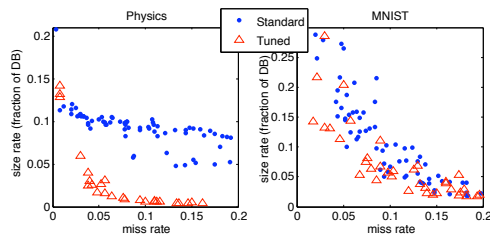

Figure 4: **Left**: Physics dataset. **Right**: MNIST dataset.

# 7 Conclusion

The primary contribution of this paper is demonstrating that building a NN search structure can be fruitfully viewed as a learning problem. We used this framework to develop algorithms that learn RCSs and KD-trees optimized for a query distribution. Possible future work includes applying the learning framework to other data structures, though we expect that even stronger results may be obtained by using this framework to develop a novel data structure from the ground up. On the theoretical side, margin-based generalization bounds may allow the use of richer classes of data structures.

### Acknowledgments

We are grateful to the NSF for support under grants IIS-0347646 and IIS-0713540. Thanks to Nikhil Rasiwasia, Sunhyoung Han, and Nuno Vasconcelos for providing the Corel50 data.

## Footnotes

[1] $\mathcal{D}_p$ is $p$-stable if for any $v \in \mathbb{R}^d$ and $Z, X_1, \ldots, X_d$ drawn iid from $\mathcal{D}_p$, $\langle v, X \rangle \overset{\mathrm{d}}{=} \|v\|_p Z$. For example, $\mathcal{N}(0, 1)$ is 2-stable.

[2] See the full version of this paper for any missing proofs.

[3]Additional experiments appear in the full version of this paper.

# References

[1] J. H. Friedman, J. L. Bentley, and R. A. Finkel. An algorithm for finding best matches in logarithmic expected time. *ACM Transactions on Mathematical Software*, 3(3):209–226, 1977.

[2] S. Arya, D. M. Mount, N. S. Netanyahu, R. Silverman, and A. Wu. An optimal algorithm for approximate nearest neighbor searching. *Journal of the ACM*, 45(6):891–923, 1998.

[3] T. Liu, A. W. Moore, A. Gray, and K. Yang. An investigation of practical approximate neighbor algorithms. In *Neural Information Processing Systems (NIPS)*, 2004.

[4] S. Dasgupta and Y. Freund. Random projection trees and low dimensional manifolds. Technical report, UCSD, 2007.

[5] P. Indyk. Nearest neighbors in high dimensional spaces. In J. E. Goodman and J. O'Rourke, editors, *Handbook of Discrete and Computational Geometry*. CRC Press, 2006.

[6] M. T. Orchard. A fast nearest-neighbor search algorithm. In *ICASSP*, pages 2297–3000, 1991.

[7] E. Vidal. An algorithm for finding nearest neighbours in (approximately) constant average time. *Pattern Recognition Letters*, 4:145–157, 1986.

[8] S. Omohundro. Five balltree construction algorithms. Technical report, ICSI, 1989.

[9] A. Beygelzimer, S. Kakade, and J. Langford. Cover trees for nearest neighbor. In *ICML*, 2006.

[10] K. L. Clarkson. Nearest-neighbor searching and metric space dimensions. In *Nearest-Neighbor Methods for Learning and Vision: Theory and Practice*, pages 15–59. MIT Press, 2006.

[11] S. Maneewongvatana and D. Mount. The analysis of a probabilistic approach to nearest neighbor searching. In *Workshop on Algorithms and Data Structures*, 2001.

[12] S. Maneewongvatana and D. Mount. Analysis of approximate nearest neighbor searching with clustered point sets. In *Workshop on Algorithm Engineering and Experimentation (ALENEX)*, 1999.

[13] Mayur Datar, Nicole Immorlica, Piotr Indyk, and Vahab S. Mirrokni. Locality-sensitive hashing scheme based on p-stable distributions. In *SCG 2004*, pages 253–262, New York, NY, USA, 2004. ACM Press.

[14] O. Bousquet, S. Boucheron, and G. Lugosi. Theory of classification: a survey of recent advances. *ESAIM: Probability and Statistics*, 9:323–375, 2004.

[15] D. Mount and S. Arya. ANN library. `http://www.cs.umd.edu/~mount/ANN/`.

[16] N. Rasiwasia, P. Moreno, and N. Vasconcelos. Bridging the gap: query by semantic example. *IEEE Transactions on Multimedia*, 2007.

